# MLLE: Modified Locally Linear Embedding Using Multiple Weights

**Zhenyue Zhang**
Department of Mathematics
Zhejiang University, Yuquan Campus,
Hangzhou, 310027, P. R. China
zyzhang@zju.edu.cn

**Jing Wang**
College of Information Science and Engineering
Huaqiao University
Quanzhou, 362021, P. R. China
Dep. of Mathematics, Zhejiang University
wroaring@yahoo.com.cn

## Abstract

The locally linear embedding (LLE) is improved by introducing multiple linearly independent local weight vectors for each neighborhood. We characterize the reconstruction weights and show the existence of the linearly independent weight vectors at each neighborhood. The modified locally linear embedding (MLLE) proposed in this paper is much stable. It can retrieve the ideal embedding if MLLE is applied on data points sampled from an isometric manifold. MLLE is also compared with the local tangent space alignment (LTSA). Numerical examples are given that show the improvement and efficiency of MLLE.

## 1   Introduction

The problem of nonlinear dimensionality reduction is to find the meaningful low-dimensional structure hidden in high dimensional data. Recently, there have been advances in developing effective and efficient algorithms to perform nonlinear dimension reduction which include isometric mapping Isomap [7], locally linear embedding (LLE) [5] and its variations, manifold charting [2], Hessian LLE [1] and local tangent space alignment (LTSA) [9]. All these algorithms cover two common steps: learn the local geometry around each data point and nonlinearly map the high dimensional data points into a lower dimensional space using the learned local information [3]. The performances of these algorithms, however, are different both in learning local information and in constructing global embedding, though each of them solves an eigenvalue problem eventually. The effectiveness of the local geometry retrieved determines the efficiency of the methods.

This paper will focus on the reconstruction weights that characterize intrinsic geometric properties of each neighborhood in LLE [5]. LLE has many applications such as image classification, image recognition, spectra reconstruction and data visualization because of its simple geometric intuitions, straightforward implementation, and global optimization [6, 11]. It is however also reported that LLE may be not stable and may produce distorted embedding if the manifold dimension is larger than one. One of the curses that make LLE fail is that the local geometry exploited by the reconstruction weights is not well-determined, since the constrained least squares (LS) problem involved for determining the local weights may be ill-conditioned. A Tikhonov regularization is generally used for the ill conditions LS problem. However, a regularized solution may be not a good approximation to the exact solution if the regularization parameter is not suitably selected.

The purpose of this paper is to improve LLE by making use of multiple local weight vectors. We will show the existence of linearly independent weight vectors that are approximately optimal. The local geometric structure determined by multiple weight vectors is much stable and hence can be used to improve the standard LLE. The modified LLE named as MLLE uses multiple weight vectors for each point in reconstruction of lower dimensional embedding. It can stably retrieve the ideal isometric

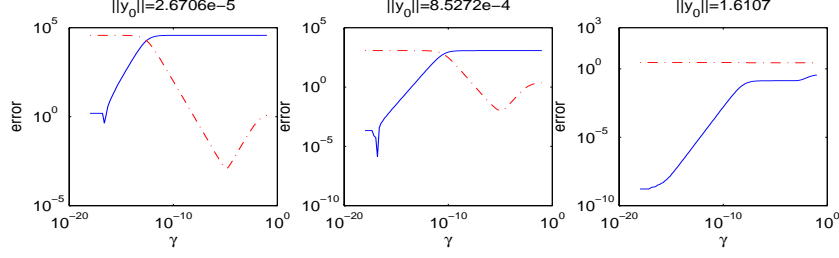

Figure 1: Examples of $\|w(\gamma) - w^*\|$ (solid line) and $\|w(\gamma) - u\|$ (dotted line) for swiss-roll data.

embedding approximately for an isometric manifold. MLLE has properties similar to LTSA both in measuring linear dependence of neighborhood and in constructing the (sparse) matrix whose smallest eigenvectors form the wanted lower dimensional embedding. It exploits the tight relations between LLE/MLLE and LTSA. Numerical examples given in this paper show the improvement and efficiency of MLLE.

## 2 The Local Combination Weights

Let $\{x_1, \ldots, x_N\}$ be a given data set of $N$ points in $\mathcal{R}^m$. LLE constructs locally linear structures at each point $x_i$ by representing $x_i$ using its selected neighbor set $\mathcal{N}_i = \{x_j, j \in J_i\}$. The optimal combination weights are determined by solving the constrained least squares problem

$$\min \|x_i - \sum_{j \in J_i} w_{ji} x_j\|, \quad s.t. \quad \sum_{j \in J_i} w_{ji} = 1. \tag{2.1}$$

Once all the reconstruction weights $\{w_{ji}, j \in J_i\}$, $i = 1, \cdots, N$, are computed, LLE maps the set $\{x_1, \ldots, x_N\}$ to $\{t_1, \ldots, t_N\}$ in a lower dimensional space $\mathcal{R}^d$ ($d < m$) that preserves the local combination properties totally,

$$\min_{T=[t_1,\ldots,t_N]} \sum_i \|t_i - \sum_{j \in J_i} w_{ji} t_j\|^2, \quad s.t. \quad TT^T = I.$$

The low dimensional embedding $T$ constructed by LLE tightly depends on the local weights. To formulate the weight vector $w_i$ consisting of the local weights $w_{ji}, j \in J_i$, let us denote matrix $G_i = [\ldots, x_j - x_i, \ldots]_{j \in J_i}$. Using the constraint $\sum_{j \in J_i} w_{ji} = 1$, we can write the combination error as $x_i - \sum_{j \in J_i} w_{ji} x_j = G_i w_i$ and hence (2.1) reads

$$\min \|G_i w\|, \quad s.t. \quad w^T \mathbf{1}_{k_i} = 1,$$

where $\mathbf{1}_{k_i}$ denotes the $k_i$-dimensional vector of all 1's. Theoretically, a null vector of $G_i$ that is not orthogonal to $\mathbf{1}_{k_i}$ can be normalized to be a weight vector as required. Otherwise, a weight vector is given by $w_i = y_i/\mathbf{1}_{k_i}^T y_i$ with $y_i$ a solution to the linear system $G_i^T G_i y = \mathbf{1}_{k_i}$ [6]. Indeed, one can formulate the solution using the singular value decomposition (SVD) of $G_i$.

**Theorem 2.1** *Let $G$ be a given matrix of $k$ column vectors. Denote by $y_0$ the orthogonal projection of $\mathbf{1}_k$ onto the null space of $G$ and $y_1 = (G^T G)^+ \mathbf{1}_k$.[1] Then the vector*

$$w^* = \frac{y^*}{\mathbf{1}_k^T y^*}, \quad y^* = \begin{cases} y_0, & y_0 \neq 0 \\ y_1, & y_0 = 0 \end{cases} \tag{2.2}$$

*is an optimal solution to $\min_{\mathbf{1}_k^T w=1} \|Gw\|$.*

The problem of solving $\min_{\mathbf{1}^T w=1} \|Gw\|$ is not stable if $G^T G$ is singular (has zero eigenvalues) or nearly singular (has relative small eigenvalues). To regularize the problem, it is suggested in [5] to solve the regularized linear system replaced

$$(G^T G + \gamma\|G\|_F^2 I)y = \mathbf{1}_k, \quad w = y/\mathbf{1}_k^T y \tag{2.3}$$

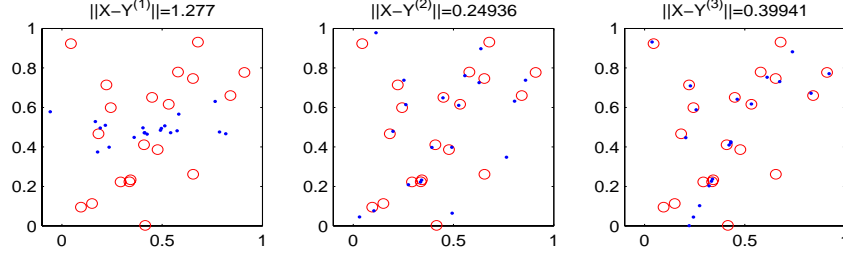

Figure 2: A 2D data set (◦-points) and computed coordinates (dot points) by LLE using different sets of optimal weight vectors (left two panels) or regularization weight vectors (right panel).

with a small positive $\gamma$. Let $y(\gamma)$ be the unique solution to the regularized linear system. One can prove that $w(\gamma) = y(\gamma)/\mathbf{1}_k^T y(\gamma)$ converges to $w^*$ as $\gamma \to 0$. However, the convergence behavior of $w(\gamma)$ is quite uncertain for small $\gamma > 0$. In fact, if $y_0 \neq 0$ is small, then $w(\gamma)$ tends to $u = \frac{y_1}{\mathbf{1}^T y_1}$ at first and then turns to the limit value $w^* = \frac{y_0}{\mathbf{1}^T y_0}$ eventually. Note that $u$ and $w^*$ are orthogonal each other. In Figure 1, we plot three examples of the error curves $\|w(\gamma) - w^*\|$ (solid line) and $\|w(\gamma) - u\|$ (dotted line) with different values of $\|y_0\|$ for the swiss-roll data. The left two panels show the metaphase phenomenon clearly, where $\|y_0\| \approx 0$. Therefore, $w^*$ can not be well approximated by $w(\gamma)$ if $\gamma$ is not small enough. This partially explains the instability of LLE.

Other factor that results in the instability of LLE is that the learned linear structure by using single weight vector at each point is brittle. LLE may give a wrong embedding even if all weight vector is well approximated in a high accuracy. It is imaginable if $G_i$ is rank reducible since multiple optimal weight vectors exist in that case. Figure 2 shows a small example of $N = 20$ two-dimensional points for which LLE fails even if exact optimal weight vectors are used. We plot three sets of computed 2D embeddings $T^{(j)}$ (within an optimal affine transformation to the ideal $X$) by LLE with $k = 4$ using two sets of exact optimal weight vectors and one set of weight vectors that solve the regularized equations, respectively. The errors $\|X - Y^{(j)}\| = \min_{c,L} \|X - (c\mathbf{1}^T + LT^{(j)})\|$ between the ideal set $X$ and the computed sets within optimal affine transformation are large in the example.

The uncertainty of $w(\gamma)$ with small $\gamma$ occurs because of existence of small singular values of $G$. Fortunately, it also implies the existence of multiple almost optimal weight vectors simultaneously. Indeed, if $G$ has $s \leq k$ small singular values, then there are $s$ approximately optimal weight vectors that are linear independent on each others. The following theorem characterizes construction of the approximately optimal weight vectors $w^{(\ell)}$ using the matrix $V$ of left singular vectors corresponding to the $s$ smallest singular values and bounds the combination errors $\|Gw^{(\ell)}\|$ in terms of the minimum of $\|Gw\|$ and the largest one of the $s$ smallest singular values.

**Theorem 2.2** *Let $G \in R^{m \times k}$ and $\sigma_1(G) \geq \ldots \geq \sigma_k(G)$ be the singular values of $G$. Denote*

$$w^{(\ell)} = (1 - \alpha)w^* + VH(:, \ell), \quad \ell = 1, \cdots, s,$$

*where $V$ is the eigenvector matrix of $G$ corresponding to the $s$ smallest right singular values, $\alpha = \frac{1}{\sqrt{s}}\|V^T \mathbf{1}_k\|$, and $H$ is a Householder matrix that satisfies $HV^T \mathbf{1}_k = \alpha \mathbf{1}_s$. Then*

$$\|Gw^{(\ell)}\| \leq \|Gw^*\| + \sigma_{k-s+1}(G). \tag{2.4}$$

The Householder matrix is symmetric and orthogonal. It is given by $H = I - 2hh^T$ with vector $h \in \mathcal{R}^s$ defined as follows. Let $h_0 = \alpha \mathbf{1}_s - V^T \mathbf{1}_k$. If $h_0 = 0$, then $h = 0$. Otherwise, $h = \frac{h_0}{\|h_0\|}$.

Note that $\|w^*\|$ can be very large when $G$ is approximately singular. In that case, $(1 - \alpha)w^*$ dominates $w^{(\ell)}$ and hence $w^{(1)}, \ldots, w^{(s)}$ are almost same and numerically linear dependent each others. Equivalently, $W = [w^{(1)}, \ldots, w^{(s)}]$ has large condition number $\text{cond}(W) = \frac{\sigma_{\max}(W)}{\sigma_{\min}(W)}$. For numerical stability, we replace $w^*$ by a regularized weight vector $w(\gamma)$ like in LLE. This modification is quite practical in application and, more importantly, it can reinforce the numerically linear independence of $\{w^{(\ell)}\}$. In our experiment, the construction of the $\{w^{(\ell)}\}$ is stable with respect to the choice of $\gamma$. We show an estimation of the condition number $\text{cond}(W)$ for the modified $W$ below.

**Theorem 2.3** *Let* $W = (1 - \alpha)w(\gamma)\mathbf{1}_s^T + VH$. *Then* $\mathrm{cond}(W) \le (1 + \sqrt{k}(1 - \alpha)\|w(\gamma)\|)^2$.

# 3 MLLE: Modified locally linear embedding

It is justifiable to learn the local structure by multiple optimal weight vectors at each point, rather than a single one. Though the exact optimal weight vector may be unique, multiple approximately optimal weight vectors exist by Theorem 2.2. We will use these weight vectors to determine an improved and more stable embedding. Below we show the details of the modified locally linear embedding using multiple local weight vectors.

Consider the neighbor set of $x_i$ with $k_i$ neighbors. Assume that the first $r_i$ singular values of $G_i$ are large compared with the remaining $s_i = k_i - r_i$ singular values. (We will discuss how to choose it later.) Let $w_i^{(1)}, \ldots, w_i^{(s_i)}$ be $s_i \le k$ linearly independent weight vectors,

$$w_i^{(\ell)} = (1 - \alpha_i)w_i(\gamma) + V_iH_i(:,\ell), \quad \ell = 1, \cdots, s_i.$$

Here $w_i(\gamma)$ is the regularized solution defined in (2.2) with $G = G_i$, $V_i$ is the matrix of $G_i$ corresponding to the $s_i$ smallest right singular values, $\alpha_i = \frac{1}{\sqrt{s_i}}\|v_i\|$ with $v_i = V_i^T\mathbf{1}_{k_i}$, and $H_i$ is a Householder matrix that satisfies $H_iV_i^T\mathbf{1}_{k_i} = \alpha_i\mathbf{1}_{s_i}$.

We look for a $d$-dimensional embedding $\{t_1, \ldots, t_N\}$, that minimizes the embedding cost function

$$E(T) = \sum_{i=1}^{N}\sum_{\ell=1}^{s_i}\|t_i - \sum_{j \in J_i} w_{ji}^{(\ell)}t_j\|^2 \tag{3.5}$$

with the constraint $TT^T = I$. Denote by $W_i = (1 - \alpha_i)w_i(\gamma)\mathbf{1}_{s_i}^T + V_iH_i$ the local weight matrix and let $\hat{W}_i \in \mathcal{R}^{N \times s_i}$ be the embedded matrix of $W_i$ into the $N$-dimensional space such that

$$\hat{W}_i(J_i, :) = W_i, \quad \hat{W}(i, :) = -\mathbf{1}_{s_i}^T, \hat{W}(j, :) = 0, \quad j \notin I_i = J_i \cup \{i\}.$$

The cost function (3.5) can be rewritten as

$$E(T) = \sum_i \|T\hat{W}_i\|_F^2 = \mathrm{Tr}(T\sum_i \hat{W}_i\hat{W}_i^TT^T) = \mathrm{Tr}(T\Phi T^T), \tag{3.6}$$

where $\Phi = \sum_i \hat{W}_i\hat{W}_i^T$. The minimizer of $E(T)$ is given by the matrix $T = [u_2, \ldots, u_{d+1}]^T$ of the $d$ eigenvectors of $\Phi$ corresponding to the 2nd to $d + 1$st smallest eigenvalues.

## 3.1 Determination of number $s_i$ of approximation optimal weight vectors

Obviously, $s_i$ should be selected such that $\sigma_{k_i - s_i + 1}(G_i)$ is relatively small. In general, if the data points are sampled from a $d$-dimensional manifold and the neighbor set is well selected, then $\sigma_d(G_i) \gg \sigma_{d+1}(G_i)$. So $s_i$ can be any integer satisfying $s_i \le k_i - d$, and $s_i = k_i - d$ is the best choice. However because of noise and that the neighborhood is possibly not well selected, $\sigma_{d+1}(G_i)$ may be not relatively small. It makes sense to choose $s_i$ as large as possible if the ratio $\frac{\lambda_{k_i-s_i+1}^{(i)} + \cdots + \lambda_{k_i}^{(i)}}{\lambda_1^{(i)} + \cdots + \lambda_{k_i-s_i}^{(i)}}$ is small, where $\lambda_j^{(i)} = \sigma_j^2(G_i)$ are the eigenvalues of $G_i^TG_i$. There is a trade between the number of weight vectors and the approximation to $\|G_iw_i^*\|$. We suggest

$$s_i = \max_{\ell}\left\{\ell \le k_i - d, \quad \frac{\sum_{j=k_i-\ell+1}^{k_i}\lambda_j^{(i)}}{\sum_{j=1}^{k_i-\ell}\lambda_j^{(i)}} < \eta\right\}, \tag{3.7}$$

for a given $\eta < 1$ that is a threshold error. Here $d$ can be over estimated to be $d' > d$.

Obviously, $s_i$ depends on the parameter $\eta$ monotonically. The smaller $\eta$ is, the smaller $s_i$ is, and of course, the smaller the combination errors for the weight vectors used are. We use an adaptive strategy to set $\eta$ as follows. Let $\rho_i = \sum_{j=d+1}^{k_i}\lambda_j^{(i)}/\sum_{j=1}^d\lambda_j^{(i)}$, $i = 1, \ldots, N$, and reorder $\{\rho_i\}$ as $\rho_{\pi_1} \le \ldots \le \rho_{\pi_N}$. Then we set $\eta$ to be the middle term of $\{\rho_i\}$, $\eta = \rho_{\pi_{\lceil N/2 \rceil}}$, where $\lceil N/2 \rceil$ is the nearest integer of $N/2$ towards infinity. In general, if the manifold near $x_i$ is float or has small

curvatures and the neighbors are well selected, $\rho_i$ is smaller than $\eta$ and $s_i = k - d$. For those neighbor sets with large local curvatures, $\rho_i > \eta$ and $s_i < k_i - d$. So less number of weight vectors are used in constructing the local linear structures and the combination errors decrease.

We summarize the Modified Locally linear Embedding (MLLE) algorithm as follows.

---

**Algorithm MLLE** (Modified Locally linear Embedding).

    **1.** For each $i = 1, \cdots, N$,

        **1.1** Determine a neighbor set $\mathcal{N}_i = \{x_j, \ j \in J_i\}$ of $x_i$, $i \notin J_i$.

        **1.2** Compute the regularized solution $w_i(\gamma)$ by (2.3) with a small $\gamma > 0$.

        **1.3** Compute the eigenvalues $\lambda_1^{(i)}, \ldots, \lambda_{k_i}^{(i)}$ and eigenvectors $v_1^{(i)}, \ldots, v_{k_i}^{(i)}$ of $G_i^T G_i$. Set
$\rho_i = \sum_{j=d+1}^{k_i} \lambda_j^{(i)} / \sum_{j=1}^{d} \lambda_j^{(i)}$.

    **2.** Sort $\{\rho_i\}$ to be $\{\rho_{\pi_i}\}$ in increasing order and set $\eta = \rho_{\pi_{\lceil N/2 \rceil}}$.

    **3.** For each $i = 1, \cdots, N$,

        **3.1** Set $s_i$ by (3.7) and set $V_i = [v_{k_i-s_i+1}^{(i)}, \ldots, v_{k_i}^{(i)}]$, $\alpha_i = \|\mathbf{1}_{k_i}^T V_i\|$.

        **3.2** Construct $\Phi$ by using $W_i = w_i(\gamma)\mathbf{1}_{s_i}^T + V_i$.

    **4.** Compute the $d + 1$ smallest eigenvectors of $\Phi$ and pick up the eigenvector matrix corresponding to the 2nd to $d + 1$st smallest eigenvalues, and set $T = [u_2, \ldots, u_{d+1}]^T$.

---

The computational cost of MLLE is almost the same as that of LLE. The additional flops of MLLE for computing the eigendecomposition of $G_i^T G_i$ is $O(k_i^3)$ and totally $O(k^3 N)$ with $k = \max_i k_i$. Note that the most computationally expensive steps in both LLE and MLLE are the neighborhood selection and the computation of the $d + 1$ eigenvectors of the alignment matrix $\Phi$ corresponding to small eigenvalues. They cost $O(mN^2)$ and $O(dN^2)$, respectively. Because $k \ll N$, the additional cost of MLLE is ignorable.

## 4 An analysis of MLLE for isometric manifolds

Consider the application of MLLE on an isometric manifold $\mathcal{M} = f(\Omega)$ with open set $\Omega \subset \mathcal{R}^d$ and smooth function $f$. Assume that $\{x_i\}$ are sampled from $\mathcal{M}$, $x_i = f(\tau_i)$, $i = 1, \ldots, N$. We have

$$\|x_i - \sum_{j \in J_i} w_{ji} x_j\| = \|\tau_i - \sum_{j \in J_i} w_{ji} \tau_j\| + O(\varepsilon_i^2), \qquad (4.8)$$

due to the isometry of $f$. If $k_i > d$, then the optimal reconstruction error of $\tau_i$ should be zero. So we have that $\|x_i - \sum_{j \in J_i} w_{ji}^* x_j\| = O(\varepsilon_i^2)$. For the approximately optimal weight vectors $w_i^{(\ell)}$, we have $\|x_i - \sum_{j \in J_i} w_{ji}^{(\ell)} x_j\| \approx \sigma_{k_i-s_i+1}(G_i) + O(\varepsilon_i^2)$. Inversely, if follows from (4.8) that $\|\tau_i - \sum_{j \in J_i} w_{ji}^{(\ell)} \tau_j\| \approx \sigma_{k_i-s_i+1}(G_i) + O(\varepsilon_i^2)$. Therefore, denoting $T^* = [\tau_1, \ldots, \tau_N]$, we have

$$E(T^*) = \sum_{i=1}^{N} \sum_{\ell=1}^{s_i} \| \sum_{j \in J_i} w_{ji}^{(\ell)} \tau_j - \tau_i \|^2 \leq \sum_{i=1}^{N} s_i \sigma_{k_i-s_i+1}^2(G_i) + O(\max_i \varepsilon_i^2).$$

For the orthogonalized $U$ of $T^*$, i.e., $T^* = LU$ and $UU^T = I$, since $L = T^* U^T \in \mathcal{R}^{d \times d}$, we have that $\sigma_d(L) = \sigma_d(T^*)$ and $E(U) \leq E(T^*)/\sigma_d^2(T^*)$. Note that $\sigma_{k_i-s_i+1}^2(G_i)$ is very small generally. So $E(U)$ is always small and approximately achieves the minimum. Roughly speaking, MLLE can retrieve the isometric embedding.

## 5 Comparison to LTSA

MLLE has similar properties similar to those of LTSA. In this section, we compare MLLE and LTSA in the linear dependence of neighbors and alignment matrices. For simplicity, we assume that $r_i = d$, i.e., $k_i - d$ weight vectors are used in MLLE for each neighbor set.

## 5.1 Linear dependence of neighbors.

The total combination error

$$\epsilon^{MLLE}(\mathcal{N}_i) = \sum_{\ell=1}^{k_i-d} \| \sum_{j \in J_i} w_{ji}^{(\ell)} x_j - x_i \|^2 = \|G_i W_i\|_F^2$$

of $x_i$ can be a measure of the linear dependence of the neighborhood $\mathcal{N}_i$. To compare it with the measure of linear dependence defined by LTSA, we denote by $\bar{x}_i = \frac{1}{|I_i|} \sum_{j \in I_i} x_j$ the mean of members in the whole neighbors of $x_i$ including $x_i$ itself, and $\bar{X}_i = [\dots, x_j - \bar{x}_i, \dots]_{j \in I_i}$. It can be verified that $G_i W_i = \bar{X}_i \tilde{W}_i$ with $\tilde{W}_i = \hat{W}_i(I_i, :)$. So $\epsilon^{MLLE}(\mathcal{N}_i) = \|\bar{X}_i \tilde{W}_i\|_F^2$.

In LTSA, the linear dependence of $\mathcal{N}_i$ is measured by the total errors

$$\epsilon^{LTSA}(\mathcal{N}_i) = \sum_{j \in I_i} \|x_j - \bar{x}_i - Q_i \theta_j^{(i)}\|^2 = \|\bar{X}_i - Q_i \Theta_i\|_F^2 = \|\bar{X}_i \tilde{V}_i\|_F^2,$$

where $\tilde{V}_i$ is the matrix consists of the right singular vectors of $\bar{X}_i$ corresponding to $k_i - d$ smallest singular values. The MLLE-measure $\epsilon^{MLLE}$ and the LTSA-measure $\epsilon^{LTSA}$ of neighborhood linear dependence are similar,

$$\epsilon^{MLLE}(\mathcal{N}_i) = \|\bar{X}_i \tilde{W}_i\|_F^2, \quad \|\bar{X}_i \tilde{w}_i^{(\ell)}\| \approx \min, \ \ell \leq k_i - d,$$
$$\epsilon^{LTSA}(\mathcal{N}_i) = \|\bar{X}_i \tilde{V}_i\|_F^2 = \min_{Z^T Z = I} \|\bar{X}_i Z\|_F^2.$$

## 5.2 Alignment matrices.

Both MLLE and LTSA minimize a trace function of an *alignment* matrix $\Phi$ to obtain an embedding, $\min_{TT^T=I} \text{trace}(T \Phi T^T)$. The alignment matrix can be written in the same form

$$\Phi = \sum_{i=1}^{N} S_i \Phi_i S_i^T,$$

where $S_i$ is a selection matrix consisting of the columns $j \in I_i$ of the large identity matrix of order $N$. In LTSA, the local matrix $\Phi_i$ is given by the orthogonal projection, i.e. $\Phi_i^{LTSA} = \tilde{V}_i \tilde{V}_i^T$, see [10]. For MLLE, $\Phi_i^{MLLE} = \tilde{W}_i \tilde{W}_i^T$. It is interesting that the range space of $\tilde{W}_i$ span$(\tilde{W}_i)$ and the range space span$(\tilde{V}_i)$ of $\tilde{V}_i$ are tightly close each other if the reconstruction error of $x_i$ is small. The following theorem gives an upper bound of the closeness using the distance dist$(\tilde{W}_i, \tilde{V}_i)$ between span$(\tilde{W}_i)$ and span$(\tilde{V}_i)$ that denotes the largest angle between the two subspaces. (See [4] for discussion about distance of subspaces.)

**Theorem 5.1** *Let* $G_i = [\cdots, x_j - x_i, \cdots]_{j \in J_i}$. *Then* dist$(\tilde{W}_i, \tilde{V}_i) \leq \frac{\|G_i W_i\|}{\sigma_d(\tilde{W}_i)\sigma_d(\bar{X}_i)}$.

# 6 Experimental Results.

In this section, we present several numerical examples to illustrate the performance of MLLE algorithm. The test data sets include simulated date sets and real world examples.

First, we compare Isomap, LLE, LTSA, and MLLE on the Swiss roll with a hole. The data points generated from a rectangle with a missing rectangle strip punched out of the center and then the resulting Swiss roll is not convex. We run these four algorithms with $k = 10$. In the top middle of Figure 3, we plot the computed coordinates by Isomap, and there is a dilation of the missing region and a warp on the rest of the embedding. As seen in the top right of Figure 3, there is a strong distortion on the computed coordinates by LLE. As we have shown in the bottom of Figure 3, LTSA and MLLE perform well.

We now compare MLLE and LTSA for a 2D manifold with 3 peaks embedded in 3D space. We generate $N = 1225$ 3D-points $x_i = [t_i, s_i, h(t_i, s_i)]^T$, where $t_i$ and $s_i$ are uniformly distributed in the interval $[-1.5, 1.5]$ and $h(t, s)$ is defined by

$$h(t, s) = e^{-10\left((t-0.5)^2+(s-0.5)^2\right)} - e^{-10\left(t^2+(s+1)^2\right)} - e^{-10\left((1+t)^2+s^2\right)}.$$

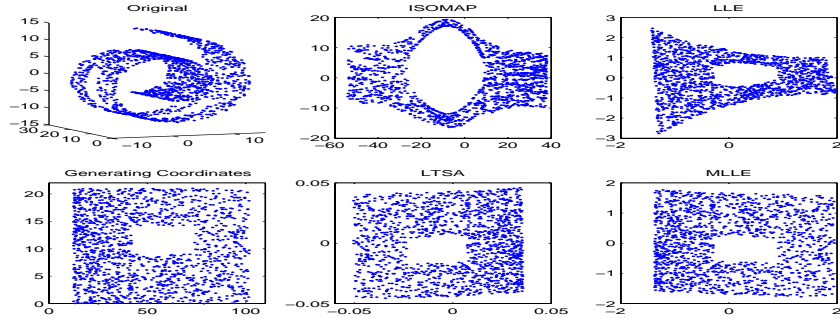

Figure 3: Left column: Swiss-roll data and generating coordinates with a missing rectangle. Middle column: computed results by Isomap and LTSA. Right column: results of LLE and MLLE.

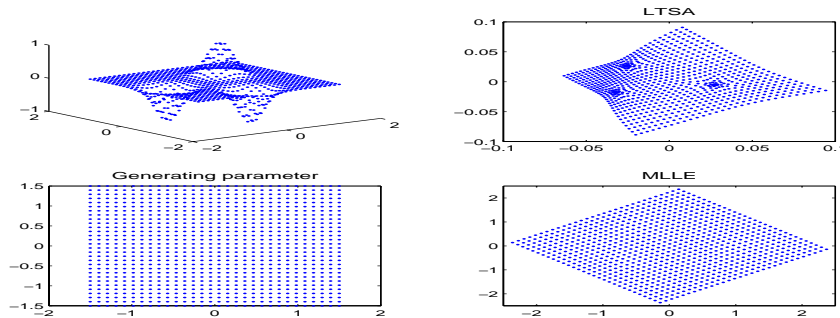

Figure 4: Left column:Plots of the 3-peak data and the generating coordinates. Right column: Results of LTSA and MLLE.

See the left of Figure 4 for the data points and the generating parameters. It is easy to show that the manifold parameterized by $f(t, s) = [t, \ s, \ h(t, s)]^T$ is approximately isometric since the Jacobian $J_f(t, s)$ is orthonormal approximately. In the right of Figure 4, we plot the computed coordinates by LTSA and MLLE with $k = 12$. The deformations of the computed coordinates by LTSA near the peaks are prominent because the curvature of the 3-peak manifold varies very much. This bias can be reduced by the modified curvature model of LTSA proposed in [8]. MLLE can recover the generating parameter perfectly up to an affine transformation.

Next, we consider a data set containing $N = 4400$ handwritten digits ('2'-'5') with 1100 examples of each class. The gray scale images of handwritten numerals are at $16 \times 16$ resolution and converted $m = 256$ dimensional vectors[2]. The data points are mapped into a 2-dimensional space using LLE and MLLE respectively. These experiments are shown in Figure 5. It is clear that MLLE performs much better than LLE. Most of the digit classes (digits '2'-'5' are marked by 'o', '$\diamond$', '$\triangleright$' and '$\triangle$' respectively) are well clustered in the resulting embedding of MLLE.

Finally, we consider application of MLLE and LLE on the real data set of 698 face images with variations of two pose parameters (left-right and up-down) and one lighting parameter. The image size is 64-by-64 pixel, and each image is converted to an $m = 4096$ dimensional vector. We apply MLLE with $k = 14$ and $d = 3$ on the data set. The first two coordinates of MLLE are plotted in the middle of Figure 6. We also extract four paths along the boundaries of the set of the first two coordinates, and display the corresponding images along each path. These components appear to capture well the pose and lighting variations in a continuous way.

## Footnotes

[1] $(\cdot)^+$ denotes the Moore-Penrose generalized inverse of a matrix.

[2]The data set can be downloaded at http://www.cs.toronto.edu/ roweis/data.html.

## References

[1] D. Donoho and C. Grimes. Hessian Eigenmaps: new tools for nonlinear dimensionality reduction. *Proceedings of National Academy of Science*, 5591-5596, 2003

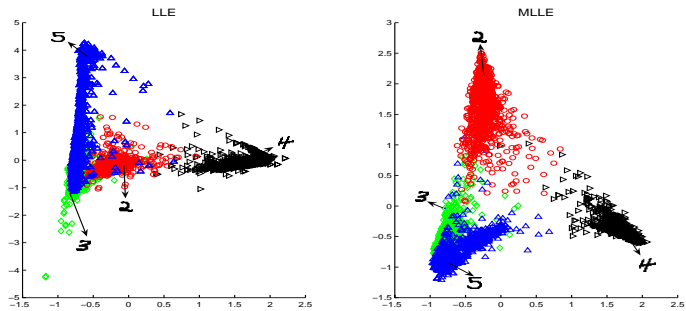

Figure 5: Embedding results of $N = 4400$ handwritten digits by LLE(left) and MLLE(right).

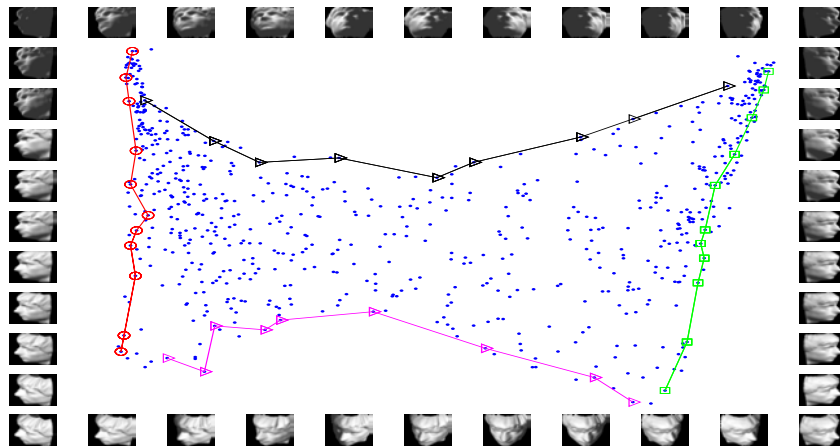

Figure 6: Images of faces mapped into the embedding described by the first two coordinates of MLLE, using the parameters $k = 14$ and $d = 3$.

[2] M. Brand. Charting a manifold. *Advances in Neural Information Processing Systems*, 15, MIT Press, 2003

[3] Jihun Ham, Daniel D. Lee, Sebastian Mika, Bernhard Scholkopf. A kernel view of the dimensionality reduction of manifolds. International Conference On Machine Learning 21, 2004.

[4] G. H. Golub and C. F Van Loan. *Matrix Computations*. Johns Hopkins University Press, Baltimore, Maryland, 3nd edition, 1996.

[5] S. Roweis and L Saul. Nonlinear dimensionality reduction by locally linear embedding. *Science*, 290: 2323–2326, 2000.

[6] L. Saul and S. Roweis. Think globally, fit locally: unsupervised learning of nonlinear manifolds. *Journal of Machine Learning Research*, 4:119-155, 2003.

[7] J Tenenbaum, V. De Silva and J. Langford. A global geometric framework for nonlinear dimension reduction. *Science*, 290:2319–2323, 2000

[8] J. Wang, Z. Zhang and H. Zha. Adaptive Manifold Learning. Advances in Neural Information Processing Systems 17, edited by Lawrence K. Saul and Yair Weiss and Léon Bottou, MIT Press, Cambridge, MA, pp.1473-1480, 2005.

[9] Z. Zhang and H. Zha. Principal Manifolds and Nonlinear Dimensionality Reduction via Tangent Space Alignment. *SIAM J. Scientific Computing*, 26(1):313–338, 2004.

[10] H. Zha and Z. Zhang. Spectral Analysis of Alignment in Manifold Learning. Submitted, 2006.

[11] M. Vlachos, C. Domeniconi, D. Gunopulos, G. Kollios, and N. Koudas Non-Linear Dimensionality Reduction Techniques for Classification and Visualization Proc. Eighth ACM SIGKDD Int'l Conf. Knowledge Discovery and Data Mining, July 2002.
